# Algorithms for Interdependent Security Games

**Michael Kearns**
**Luis E. Ortiz**
Department of Computer and Information Science
University of Pennsylvania

## 1 Introduction

Inspired by events ranging from 9/11 to the collapse of the accounting firm Arthur Andersen, economists Kunreuther and Heal [5] recently introduced an interesting game-theoretic model for problems of *interdependent security (IDS)*, in which a large number of players must make individual investment decisions related to security — whether physical, financial, medical, or some other type — but in which the ultimate safety of each participant may depend in a complex way on the actions of the entire population. A simple example is the choice of whether to install a fire sprinkler system in an individual condominium in a large building. While such a system might greatly reduce the chances of the owner's property being destroyed by a fire originating *within* their own unit, it might do little or nothing to reduce the chances of damage caused by fires originating in *other* units (since sprinklers can usually only douse small fires early). If "enough" other unit owners have not made the investment in sprinklers, it may be not cost-effective for any individual to do so.

Kunreuther and Heal [5] observe that a great variety of natural problems share this basic interdependent structure, including investment decisions in airline baggage security (in which investments in new screening procedures may reduce the risk of directly checking suspicious cargo, but nearly all airlines accept transferred bags with no additional screening [1]); risk management in corporations (in which individual business units have an incentive to avoid high-risk or illegal activities only if enough other units are similarly well-behaved); vaccination against infectious disease (where the fraction of the population choosing vaccination determines the need for or effectiveness of vaccination); certain problems in computer network security; and many others. All these problems share the following important properties:

- There is a "bad event" (condominium fire, airline explosion, corporate bankruptcy, infection, etc.) to be avoided, and the opportunity to reduce the risk of it via some kind of investment.

- The cost-effectiveness of the security investment for the individual is a function of the investment decisions made by the others in the population.

The original work by Kunreuther and Heal [5] proposed a parametric game-theoretic model for such problems, but left the interesting question of *computing* the equilibria of model largely untouched. In this paper we examine such computational issues.

## 2 Definitions

In an *IDS game*, each player $i$ must decide whether or not to invest in some abstract security mechanism or procedure that can reduce their risk of experiencing some abstract bad event. The cost of the investment to $i$ is $C_i$, while the cost of experiencing the bad event is $L_i$; the interesting case is when $L_i >> C_i$. Thus, player $i$ has two choices for his action $a_i$: $a_i = 1$ means the player makes the investment, while $a_i = 0$ means he does not. It turns out that the important parameter is the *ratio* of the two costs, so we define $R_i = C_i/L_i$.

For each player $i$, there is a parameter $p_i$, which is the probability that player $i$ will experience the bad event due to *internal* contamination if $a_i = 0$ — for example, this is the probability of the condominium owner's unit burning down *due to a fire originating in his own unit*. We can also think of $p_i$ as a measure of the *direct* risk to player $i$ — as we shall see, it is that portion of his risk under his direct control.

To model sources of *indirect* risk, for each *pair* of players $i, j, i \neq j$, let $q_{ji}$ be the probability that player $i$ experiences the bad event as a result of a *transfer* from player $j$ — for example, this is the probability that the condominium of player $i$ burns down due to a fire originating in the unit of player $j$. Note the implicit constraint that $p_i + \sum_{j \neq i} q_{ji} < 1$.

An IDS game is thus given by the parameters $p_i$, $q_{ji}$, $L_i$, $C_i$ for each player $i$, and the expected cost to player $i$ under the model is defined to be

$$M_i(\vec{a}) = a_i C_i + (1-a_i)p_i L_i + (1 - (1-a_i)p_i) \left[ 1 - \prod_{j=1, j \neq i}^{n} (1 - (1-a_j)q_{ji}) \right] L_i \quad (1)$$

Let us take a moment to parse and motivate this definition, which is the sum of three terms. The first term represents the amount invested in security by player $i$, and is either 0 (if $a_i = 0$) or $C_i$ (if $a_i = 1$). The second term is the expected cost to $i$ due to internal or direct risk of the bad event, and is either $p_i L_i$ (which is the expected cost of internally generated bad events in the case $a_i = 0$), or is 0 (in the case of investment, $a_i = 1$). Thus, there is a natural tension between the first two terms: players can either invest in security, which costs money but reduces risk, or gamble by not investing. Note that here we have assumed that security investment *perfectly* eradicates direct risk (but not indirect risk); generalizations are obviously possible, but have no qualitative effect on the model.

It is the third term of Equation (1) that expresses the *interdependent* nature of the problem. This term encodes the assumption that there are $n$ sources of risk to player $i$ — his own internal risk, and a specific transfer risk from each of the other $n-1$ players — and that all these sources are statistically independent. The prefactor $(1 - (1-a_i)p_i)$ is simply the probability that player $i$ does *not* experience the bad event due to direct risk. The bracketed expression is the probability that player $i$ experiences a bad event due to transferred risk: each factor $(1 - (1-a_j)q_{ji})$ in the product is the probability that a bad event does *not* befall player $i$ due to player $j$ (and the product expresses the assumption that all of these possible transfer events are independent). Thus 1 minus this product is the probability of transferred contamination, and of course the product of the various risk probabilities is also multiplied by the cost $L_i$ of the bad event.

The model parameters and Equation (1) define a compact representation for a multiplayer game in which each player's goal is to minimize their cost. Our interest is in the efficient computation of Nash equilibria (NE) of such games [2].

# 3 Algorithms

We begin with the observation that it is in fact computationally straightforward to find a single *pure* NE of any IDS game. To see this, it is easily verified that if there are *any* conditions under which player $i$ prefers investing ($a_i = 1$) to not investing ($a_i = 0$) according to the expected costs given by Equation (1), then it is certainly the case that $i$ will prefer to invest when all the other $n - 1$ players are doing so. Similarly, the most favorable conditions for not investing occur when no other players are investing. Thus, to find a pure NE, we can first check whether either all players investing, or no players investing, forms a NE. If so, we are finished. If neither of these extremes are a NE, then there are some players for whom investing or not investing is a dominant strategy (a best response independent of the behavior of others). If we then "clamp" such players to their dominant strategies, we obtain a new IDS game with fewer players (only those without dominant strategies in the original game), and can again see if this modified game has any players with dominant strategies. At each stage of this iterative process we maintain the invariant that clamped players are playing a best response to *any* possible setting of the unclamped players.

**Theorem 1** *A pure NE for any $n$-player IDS game can be computed in time $O(n^2)$.*

In a sense, the argument above demonstrates the fact that in most "interesting" IDS games (those in which each player is a true participant, and can have their behavior swayed by that of the overall population), there are two trivial pure NE (all invest and none invest). However, we are also interested in finding NE in which some players are choosing to invest and others not to (even though no player has a dominant strategy). A primary motivation for finding such NE is the appearance of such behavior in "real world" IDS settings, where individual parties do truly seem to make differing security investment choices (such as with sprinkler systems in large apartment buildings). Conceptually, the most straightforward way to discover such NE would be to compute *all* NE of the IDS game. As we shall eventually see, for computational efficiency such a demand requires restrictions on the parameters of the game, one natural example of which we now investigate.

## 3.1 Uniform Transfer IDS Games

A *uniform transfer* IDS game is one in which the transfer risks *emanating from* a given player are independent of the transfer destination. Thus, for any player $j$, we have that for all $i \neq j$, $q_{ji} = \delta_j$ for some value $\delta_j$. Note that the risk level $\delta_j$ presented to the population by different players $j$ may still vary with $j$ — but each player spreads their risk indiscriminately across the rest of the population. An example would be the assumption that each airline transferred bags with equal probability to all other airlines.

In this section, we describe two different approaches for computing NE in uniform transfer IDS games. The first approach views a uniform transfer IDS game as a special type of *summarization game*, a class recently investigated by Kearns and Mansour [4]. In an $n$-player summarization game, the payoff of each player $i$ is a function of the actions $\vec{a}_{-i}$ of all the other players, but only *through* the value of a global and common real-valued *summarization function* $\mathcal{S}(\vec{a})$. The main result of [4] gives an algorithm for computing approximate NE of summarization games, in which the quality of the approximation depends on the *influence* of the summarization function $\mathcal{S}$. A well-known notion in discrete functional analysis, the influence of $\mathcal{S}$ is the maximum change in $\mathcal{S}$ that any input (player) can unilaterally cause. (See [4] for detailed definitions.)

It can be shown (details omitted) that any uniform transfer IDS game is in fact a summa-

rization game under the choice

$$\mathcal{S}(\vec{a}) = \prod_{j=1}^{n} (1 - (1 - a_j)\delta_j) \tag{2}$$

and that the influence of this function is bounded by the largest $\delta_j$. We note that in many natural uniform transfer IDS settings, we expect this influence to diminish like $1/n$ with the number of players $n$. (This would be the case if the risk transfer comes about through physical objects like airline baggage, where each transfer event can have only a single destination.) Combined with the results of [4], the above discussion can be shown to yield the following result.

**Theorem 2** *There is an algorithm that takes as input any uniform transfer IDS game, and any $\epsilon > 0$, and computes an $O(\epsilon + \tau\rho)$-NE, where $\rho = \max_j\{(1 - p_j)/(1 - \delta_j)\}$ and $\tau = \max_j\{\delta_j\}$. The running time of the algorithm is polynomial in $n$, $1/\epsilon$, and $\rho$.*

We note that in typical IDS settings we expect both the $p_j$ and $\delta_j$ to be small (the bad event is relatively rare, regardless of its source), in which case $\rho$ may be viewed as a constant. Furthermore, it can be verified that this algorithm will in fact be able to compute approximate NE in which some players choose to invest and others not to, even in the absence of any dominant strategies.

While viewing uniform transfer IDS games as bounded influence summarization games relates them to a standard class and yields a natural approximation algorithm, an improved approach is possible. We now present an algorithm (Algorithm **UniformTransferIDSNash** in Figure 3.1) that efficiently computes *all* NE for uniform transfer IDS games. The algorithm (indeed, even the representation of certain NE) requires the ability to compute $m$th roots.

We may assume without loss of generality that for all players $i$, $\delta_i > 0$, and $p_i > 0$. For a joint mixed strategy vector $\vec{x} \in [0, 1]^n$, denote the set of *(fully) investing players* as $I \equiv \{i : x_i = 1\}$, the set of *(fully) non-investing players* as $N \equiv \{i : x_i = 0\}$, and the set of *partially investing players* as $P \equiv \{i : 0 < x_i < 1\}$.

The correctness of algorithm **UniformTransferIDSNash** follows immediately from two lemmas that we now state without proof due to space considerations. The first lemma is a generalization of Proposition 2 of [2], and essentially establishes that the values $R_i/p_i$ and $(1-\delta_i)R_i/p_i$ determine a two-level ordering of the players' willingness to invest. This double ordering generates the outer and inner loops of algorithm **UniformTransferIDSNash**. Note that a player with small $R_i/p_i$ has a combination of relatively low cost of investing compared to the loss of a bad event (recall $R_i = C_i/L_i$), and relatively high direct risk $p_i$, and thus intuitively should be more willing to invest than players with large $R_i/p_i$. The lemma makes this intuition precise.

**Lemma 3** *(Ordering Lemma) Let $\vec{x}$ be a NE for a uniform transfer IDS game $\mathcal{G} = (n, \vec{R}, \vec{p}, \vec{\delta})$. Then for any $i \in I$ (an investing player), any $j \in N$ (a partially investing player), and any $k \in P$ (a non-investing player), the following conditions hold:*

$$
\begin{aligned}
R_i/p_i &< R_j/p_j \\
R_i/p_i &\leq (1 - \delta_k)\,R_k/p_k < R_k/p_k \\
(1 - \delta_j)\,R_j/p_j &< (1 - \delta_k)\,R_k/p_k
\end{aligned}
$$

The second lemma establishes that if a NE contains some partially investing players, the values for their mixed strategies is in fact uniquely determined. The equations for these mixed strategies is exploited in the subroutine **TestNash**.

---

Algorithm **UniformTransferIDSNash**
**Input:** An $n$-player uniform transfer IDS game $\mathcal{G}$ with direct risk parameters $\vec{p}$, transfer risk parameters $\vec{\delta}$, and cost parameters $\vec{R}$, where $R_i = C_i/L_i$.
**Output:** A set $S$ of *all exact connected sets of NE* for $\mathcal{G}$.

1. Initialize a partition of the players into three sets $I$, $N$, $P$ (the investing, not investing, and partially investing players, respectively) and test if everybody investing is a NE:
   $I \leftarrow \{1, \ldots, n\}; N \leftarrow \emptyset; P \leftarrow \emptyset; S \leftarrow$ **TestNash**$(\mathcal{G}, I, N, P, S)$

2. Let $(i_1, i_2, ..., i_n)$ be an ordering of the $n$ players satisfying $R_{i_1}/p_{i_1} \geq \ldots \geq R_{i_n}/p_{i_n}$. Call this the *outer ordering*.

3. **for** $k = 1, \ldots, n$

   (a) Move the next player in the outer ordering from the investing to the partially-investing sets: $P \leftarrow P \bigcup \{i_k\}; I \leftarrow I - \{i_k\}$

   (b) Let $(j_1, ..., j_k)$ be an ordering of the players in $P$ satisfying $(1-\delta_{j_1}) \, R_{j_1}/p_{j_1} \geq \ldots \geq (1 - \delta_{j_k}) \, R_{j_k}/p_{j_k}$. Call this the *inner ordering*.

   (c) Consider a strategy with no not-investing players: $N \leftarrow \emptyset; S \leftarrow$ **TestNash**$(\mathcal{G}, I, N, P, S)$

   (d) **for** $m = 1, \ldots, k$

      i. Move the next player in the inner ordering from the partially-investing to non-investing sets, and test if there is a NE consistent with the partition: $N \leftarrow N \bigcup \{j_m\}; P \leftarrow P - \{j_m\}; S \leftarrow$ **TestNash**$(\mathcal{G}, I, N, P, S)$

Subroutine **TestNash**
**Inputs:** An $n$-player uniform transfer IDS game $\mathcal{G}$; a partition of the players $I$, $N$, $P$ (as above); $S$, the current discovered set of connected sets of NE for $\mathcal{G}$
**Output:** $S$ with possibly one additional connected set of NE of $\mathcal{G}$ consistent with $I$, $N$, and $P$ (assuming unit-time computation of $m$-roots of rational numbers)

1. Set pure strategies for not-investing and investing players, respectively: $\forall k \in N, x_k \leftarrow 0, \forall i \in I, x_i \leftarrow 1$.

2. **if** $|P| = 1$ (Lemma 4, part (a) applies)

   (a) Let $P = \{j\}$, $U$ as in Equation 3 and $U' = U \bigcap (0, 1)$

   (b) **if** $R_j = p_j \prod_{k \in N}(1 - \delta_k)$ (i.e., player $j$ is indifferent) and $U' \neq \emptyset$, **then** return $S \bigcup \{\{\vec{y} : y_j \in U', \vec{y}_{-j} = \vec{x}_{-j}\}\}$

3. **else** (Lemma 4, part (b) applies)

   (a) Compute mixed strategies $\forall j \in P$, $x_j$ as in Equation 4

   (b) **if** $\exists j \in P, x_j \leq 0$ or $x_j \geq 1$, return $S$

   (c) **if** $\vec{x}$ is a NE for $\mathcal{G}$ **then** return $S \bigcup \{\{\vec{x}\}\}$

4. return $S$

---

Figure 1: Algorithm **UniformTransferIDSNash**

If $I = [l, u]$ is an interval of $\Re$ with endpoints $l$ and $u$, and $a, b \in \Re$ then we define $aI + b \equiv [al + b, au + b]$.

**Lemma 4** *(Partial Investment Lemma) Let $\vec{x} \in [0, 1]^n$ be a mixed strategy for a uniform transfer IDS game $\mathcal{G} = (n, \vec{R}, \vec{p}, \vec{\delta})$, and let $P$ be the set of partially investing players in $\vec{x}$. Then (a) if $|P| = 1$, then letting $P = \{j\}$, $V = [\max_{i \in I} R_i/p_i, \; \min_{k \in N}(1 - \delta_k) \, R_k/p_k]$, and*

$$U = ((p_j/R_j) \, V - (1 - \delta_j)) \, / \, \delta_j \qquad (3)$$

*it holds that $\vec{x}$ is a NE if and only if $R_j = p_j \prod_{k \in N} 1 - \delta_k$ (i.e., player $j$ is indifferent) and player $j$ mixed strategy satisfies $x_j \in U$; else, (b) if $|P| > 1$, and $\vec{x}$ is a NE, then for all*

$j \in P$,

$$x_j = \left( (p_j/R_j)E - (1 - \delta_j) \right) / \delta_j \tag{4}$$

*where* $E = \left( \prod_{j \in P} (R_j/p_j) \Big/ \prod_{k \in N} (1 - \delta_k) \right)^{1/(|P|-1)}$ .

The next theorem summarizes our second algorithmic result on uniform transfer IDS games. The omitted proof follows from Lemmas 3 and 4.

**Theorem 5** *Algorithm* **UniformTransferIDSNash** *computes all exact (connected sets of) NE for uniform transfer IDS games in time polynomial in the size of the model.*

We note that it follows immediately from the description and correctness of the algorithm that any $n$-player uniform transfer IDS game has at most $n(n+3)/2 + 1$ connected sets of NE. In addition, each connected set of NE in a uniform transfer IDS game is either a singleton or a simple interval where $n - 1$ of the players play pure strategies and the remaining player has a simple interval in $[0, 1]$ of probability values from which to choose its strategy. At most $n$ of the connected sets of NE in a uniform transfer IDS game are simple intervals.

### 3.2 Hardness of General IDS Games

In light of the results of the preceding section, it is of course natural to consider the computational difficulty of unrestricted IDS. We now show that even a slight generalization of uniform transfer IDS games, in which we allow the $\delta_j$ to assume two fixed values instead of one, leads to the intractabilty of computing at least some of the NE.

A *graphical uniform transfer IDS game*, so named because it can be viewed as a marriage between uniform transfer IDS games and the graphical games introduced in [3], is an IDS game with the restriction that for all players $j$, $q_{ji} \in \{0, \delta_j\}$, for some $\delta_j > 0$. Let $N(j) \equiv \{i : q_{ji} > 0\}$ be the set of players that can be *directly affected* by player $j$'s behavior. In other words, the transfer risk parameter $q_{ji}$ of player $j$ with respect to player $i$ is either zero, in which case the player $j$ has no *direct* effect on player $i$'s behavior; or it is constant, in which case, the public safety $e_{ji} = (1 - (1 - x_j)\delta_j)$ of player $j$ with respect to player $i \in N(j)$ is the same as for any other player in $N(j)$.

The *pure Nash extension problem* for an $n$-player game with binary actions takes as input a description of the game and a partial assignment $\vec{a} \in \{0, 1, *\}^n$. The output may be any complete assignment (joint action) $\vec{b} \in \{0, 1\}^n$ that agrees with $\vec{a}$ on all its 0 and 1 settings, and is a (pure) NE for the game; or "none" if no such NE exists. Clearly the problem of computing *all* the NE is at least as difficult as the pure Nash extension problem.

**Theorem 6** *The pure Nash extension problem for graphical uniform transfer IDS games is NP-complete, even if $|N(j)| \leq 3$ for all $j$, and $\delta_j$ is some fixed value $\delta$ for all $j$.*

The reduction (omitted) is from Monotone One-in-Three SAT [1].

## 4 Experimental Study: Airline Baggage Security

As an empirical demonstration of IDS games, we constructed and conducted experiments on an IDS game for airline security that is based on real industry data. We have access to a data set consisting of 35,362 records of actual civilian commercial flight reservations, both domestic and international, made on August 26, 2002. Since these records contain complete flight itineraries, they include passenger transfers between the 122 represented commercial air carriers. As described below, we used this data set to construct an IDS

game in which the players are the 122 carriers, the "bad event" corresponds to a bomb exploding in a bag being transported in a carrier's airplane, and the transfer event is the physical transfer of a bag from one carrier to another.

For each carrier pair $(i, j)$, the transfer parameter $q_{ji}$ was set to be proportional to the count of transfers from carrier $j$ to carrier $i$ in the data set. We are thus using the rate of *passenger* transfers as a proxy for the rate of *baggage* transfers. The resulting parameters (details omitted) are, as expected, quite asymmetric, as there are highly structured patterns of transfers resulting from differing geographic coverages, alliances between carriers, etc. The model is thus far from being a uniform transfer IDS game, and thus algorithm **UniformTransferIDSNash** cannot be applied; we instead used a simple gradient learning approach.

The data set provides no guidance on reasonable values for the $R_i$ and $p_i$, which quantify relative costs of a hypothetical new screening procedure and the direct risks of checking contaminated luggage, respectively; presumably $R_i$ depends on the specific economics of the carrier, and $p_i$ on some notion of the risk presented by the carrier's clientele, which might depend on the geographic area served. Thus, for illustrative purposes, an arbitrary value of $p_i = 0.01$ was chosen for all $i$ [3], and a common value for $R_i$ of 0.009 (so an explosion is roughly 110 times more costly to a carrier than full investment in security).

Since the asymmetries of the $q_{ji}$ preclude the use of algorithm **UniformTransferIDSNash**, we instead used a learning approach in which each player begins with a random initial investment strategy $x_i \in [0, 1]$, and adjusts its degree of investment up or down based on the gradient dynamics $x_i \leftarrow x_i - \eta \Delta_i$, where $\Delta_i$ is determined by computing the derivative of Equation (1) and $\eta = 0.05$ was used in the experiments to be discussed.

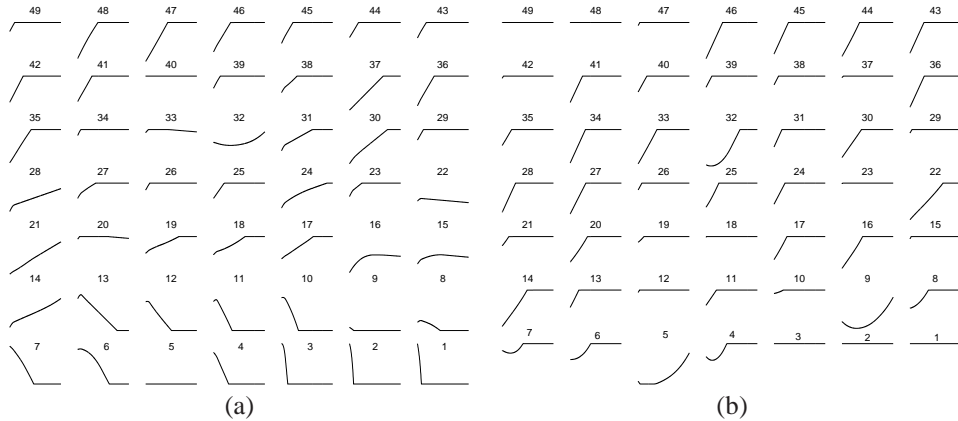

(a)                    (b)

Figure 2: (a) Simulation of the evolution of security investment strategies for the 49 busiest carrier using gradient dynamics under the IDS model. Above each plot is an index indicating the rank of the carrier in terms of overall volume in the data set. Each plot shows the investment level $x_i$ (initialized randomly in $[0, 1]$) for carrier $i$ over 500 simulation steps. (b) Tipping phenomena. Simulation of the evolution of security investment strategies for the 49 busiest carriers, but with the three largest carriers (indices 1, 2 and 3) in the data set clamped (subsidized) at full investment. The plots are ordered as in (a), and again show 500 simulation steps under gradient dynamics.

Figure 2(a) shows the evolution, over 500 steps of simulation time, of the investment level $x_i$ for the 49 busiest carriers [4]. We have ordered the 49 plots with the least busy carrier

[3]This is (hopefully) an unrealistically large value for the real world; however, it is the relationship between the parameters and not their absolute magnitudes that is important in the model.

[4]According to the total volume of flights per carrier in the data set.

(index 49) plotted in the upper left corner, and the busiest (index 1) in the lower right corner. The horizontal axes measure the 500 time steps, while the vertical axes go from 0 to 1. The axes are unlabeled for legibility.

The most striking feature of the figure is the change in the evolution of the investment strategy as we move from less busy to more busy carriers. Broadly speaking, there is a large population of lower-volume carriers (indices 49 down to 34) that quickly converge to full investment ($x_i = 1$) regardless of initial conditions. The smallest carriers, not shown (ranks 122 down to 50), also all rapidly converge to full investment. There is then a set of medium-volume carriers whose limiting strategy is approached more slowly, and may eventually converge to either full or no investment (roughly indices 33 down to 14). Finally, the largest carriers (indices 13 and lower) again converge quickly, but to no investment ($x_i = 0$), because they have a high probability of having bags transferred from other carriers (even if they protect themselves against dangerous bags being loaded directly on their planes).

Note also that the dynamics can yield complex, nonlinear behavior that includes reversals of strategy. The simulation eventually converges (within 2000 steps) to a (Nash) equilibrium in which some carriers are at full investment, and the rest at no investment. This property is extremely robust across initial conditions and model parameters,

The above simulation model enables one to examine how subsidizing several airlines to encourage it to invest in security can encourage others to do the same. This type of "tipping" behavior [6] can be the basis for developing strategies for inducing adoption of security measures short of formal regulations or requirements. Figure2(b) shows the result of an identical simulation to the one discussed above, except the three largest carriers (indices 1, 2 and 3) are now "clamped" or forced to be at full investment during the entire simulation. Independent of initial conditions, the remaining population now invariably converges to full investment. Thus the model suggests that these three carriers form (one of perhaps many different) tipping sets — carriers whose decision to invest (due to subsidization or other exogenous forces) will create the economic incentive for a large population of otherwise skeptical carriers to follow. The dynamics also reveal a cascading effect — for example, carrier 5 moves towards full investment (after having settled comfortably at no investment) only after a number of larger and smaller carriers have done so.

**Acknowledgements:** We give warm thanks to Howard Kunreuther, Geoffrey Heal and Kilian Weinberger for many helpful discussions.

## Footnotes

[1] El Al airlines is the exception to this.

[2]See (for example) [4] for definitions of Nash and approximate Nash equilibria.

## References

[1] Michael Garey and David Johnson. *Computers and Intractability: A Guide to the Theory of NP-completeness*. Freeman, 1979.

[2] Geoffrey Heal and Howard Kunreuther. You only die once: Managing discrete interdependent risks. 2003. Working paper, Columbia Business School and Wharton Risk Management and Decision Processes Center.

[3] M. Kearns, M. Littman, and S. Singh. Graphical models for game theory. In *Proceedings of the Conference on Uncertainty in Artificial Intelligence*, pages 253–260, 2001.

[4] M. Kearns and Y. Mansour. Efficient Nash computation in summarization games with bounded influence. In *Proceedings of the Conference on Uncertainty in Artificial Intelligence*, 2002.

[5] Howard Kunreuther and Geoffrey Heal. Interdependent security. *Journal of Risk and Uncertainty (Special Issue on Terrorist Risks)*, 2003. In press.

[6] Thomas Schelling. *Micromotives and Macrobehavior*. Norton, 1978.